# Dynamic Modulation of Neurons and Networks

Eve Marder
Center for Complex Systems
Brandeis University
Waltham, MA 02254 USA

## Abstract

Biological neurons have a variety of intrinsic properties because of the large number of voltage dependent currents that control their activity. Neuromodulatory substances modify both the balance of conductances that determine intrinsic properties and the strength of synapses. These mechanisms alter circuit dynamics, and suggest that functional circuits exist only in the modulatory environment in which they operate.

## 1 INTRODUCTION

Many studies of artificial neural networks employ model neurons and synapses that are considerably simpler than their biological counterparts. A variety of motivations underly the use of simple models for neurons and synapses in artificial neural networks. Here, I discuss some of the properties of biological neurons and networks that are lost in overly simplified models of neurons and synapses. A fundamental principle in biological nervous systems is that neurons and networks operate over a wide range of time scales, and that these are modified by neuromodulatory substances. The flexible, multiple time scales in the nervous system allow smooth transitions between different modes of circuit operation.

## 2 NEURONS HAVE DIFFERENT INTRINSIC PROPERTIES

Each neuron has complex dynamical properties that depend on the number and kind of ion channels in its membrane. Ion channels have characteristic kinetics and voltage

dependencies that depend on the sequence of amino acids of the protein. Ion channels may open and close in several milliseconds; others may stay open for hundreds of milliseconds or several seconds.

Some neurons are *silent* unless they receive synaptic inputs. Silent neurons can be activated by depolarizing synaptic inputs, and many will fire on rebound from a hyperpolarizing input (postinhibitory rebound). Some neurons are *tonically* active in the absence of synaptic inputs, and synaptic inputs will increase or decrease their firing rate.

Some neurons display rhythmic bursts of action potentials. These *bursting* neurons can display stable patterns of oscillatory activity, that respond to perturbing stimuli with behavior characteristic of oscillators, in that their period can be stably reset and entrained. Bursting neurons display a number of different voltage and time dependent conductances that interact to produce slow membrane potential oscillations with rapid action potentials riding on the depolarized phase. In a neuron such as R15 of *Aplysia* (Adams and Levitan 1985) or the AB neuron of the stomatogastric ganglion (STG) (Harris-Warrick and Flamm 1987), the time scale of the burst is in the second range, but the individual action potentials are produced in the 5-10msec time scale.

Neurons can generate bursts by combining a variety of different conductances. The particular balance of these conductances can have significant impact on the oscillator's behavior (Epstein and Marder 1990; Kepler *et al 1990;* Skinner *et al* 1993), and therefore the choice of oscillator model to use must be made with care (Somers and Kopell 1993).

Some neurons have a balance of conductances that give them bistable membrane potentials, allowing to produce *plateau potentials*. Typically, such neurons have two relatively stable states, a hyperpolarized silent state, and a sustained depolarized state in which they fire action potentials. The transition between these two modes of activity can be made with a short depolarizing or hyperpolarizing pulse (Fig. 1). Plateau potentials, like "flip-flops" in electronics, are a "short-term memory" mechanism for neural circuits.

The intrinsic properties of neurons can be modified by sustained changes in membrane potential. Because the intrinsic properties of neurons depend on the balance of conductances that activate and inactivate in different membrane potential ranges and over a variety of time scales, hyperpolarization or depolarization can switch a neuron between modes of intrinsic activity (Llinás 1988; McCormick 1991; Leresche *et al* 1991).

An interesting "memory-like" effect is produced by the slow inactivation properties of some $K^+$ currents (McCormick 1991; Storm 1987). In cells with such currents a sustained depolarization can "amplify" a synaptic input from subthreshold to suprathreshold, as the sustained depolarization causes the $K^+$ current to inactivate (Marom and Abbott 1994; Turrigiano, Marder and Abbott in preparation). This is another "short-term memory" mechanism that does not depend on changes in synaptic efficacy.

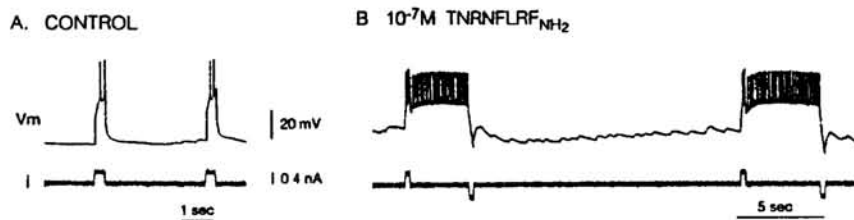

Figure 1: Intracellular recording from the DG neuron of the crab STG. A: control saline, a depolarizing current pulse elicits action potentials for its duration. B: In SDRNFLRFamide, a short depolarization elicits a plateau potential that lasts until a short hyperpolarizing current pulse terminates it. Modified from Weimann *et al* 1993.

## 2   INTRINSIC MEMBRANE PROPERTIES ARE MODULATED

Biological nervous systems use many substances as neurotransmitters and neuromodulators. The effects of these substances include opening of rapid, relatively non-voltage dependent ion channels, such as those mediating conventional rapid synaptic potentials. Alternatively, modulatory substances can change the number or type of voltage-dependent conductances displayed by a neuron, and in so doing dramatically modify the intrinsic properties of a neuron. In Fig. 1, a peptide, SDRNFLRFamide transforms the DG neuron of the crab STG from a state in which it fires only during a depolarizing pulse to one in which it displays long-lasting plateau properties (Weimann *et al* 1993). The salient feature here is that modulatory substances can elicit slow membrane properties not otherwise expressed.

## 3   SYNAPTIC STRENGTH IS MODULATED

In most neural network models synaptic weights are modified by learning rules, but are not dependent on the temporal pattern of presynaptic activity. In contrast, in many biological synapses the amount of transmitter released depends on the frequency of firing of the presynaptic neuron. Facilitation, the increase in the amplitude of the postsynaptic current when the presynaptic neuron is activated several times in quick succession is quite common. Other synapses show depression. The same neuron may show facilitation at some of its terminals while showing depression at others (Katz *et al* 1993). The facilitation and depression properties of any given synapse can not be deduced on first principles, but must be determined empirically.

Synaptic efficacy is often modified by modulatory substances. A dramatic example is seen in the *Aplysia* gill withdrawal reflex, where serotonin significantly enhances the amplitude of the monosynaptic connection from the sensory to motor neurons (Clark and Kandel 1993; Emptage and Carew 1993). The effects of modulatory substances can occur on different branches on a neuron independently (Clark and Kandel 1993), and the same modulatory substance may have different actions at different sites of the same neuron.

Electrical synapses are also subject to neuromodulation (Dowling, 1989). For example, in the retina dopamine reversibly uncouples horizonal cells.

Modulation of synaptic strength can be quite extreme; in some cases synaptic contacts may be virtually invisible in some modulatory environments, while strong in others. The implications of this for circuit operation will be discussed below.

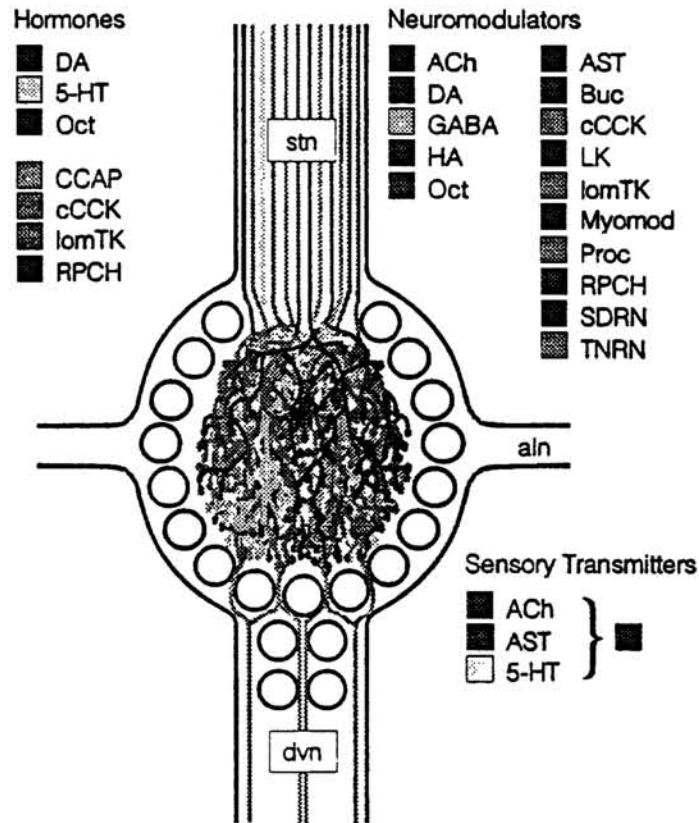

Figure 2: Modulatory substances found in inputs to the STG. See Harris-Warrick *et al.*, 1992 for details. Figure courtesy of P. Skiebe.

## 4  TRANSMITTERS ARE COLOCALIZED IN NEURONS

The time course of a synaptic potential evoked by a neurotransmitter or modulator is a characteristic property of the ion channels gated by the transmitter and/or the second messenger system activated by the signalling molecule. Synaptic currents can be relatively fast, such as the rapid action of ACh at the vertebrate skeletal neuromuscular junction where the synaptic currents decay in several milliseconds. Alternatively, second messenger activated synaptic events may have durations lasting hundreds of milliseconds, seconds, or even minutes. Many neurons contain several different neurotransmitters. It is common to find a small molecule such as glutamate or GABA colocalized with an amine such as serotonin or histamine and one or more neuropeptides. To describe the synaptic actions of such neurons, it is necessary to determine for each signalling molecule how its release depends on the frequency and pattern of activity in the presynaptic

terminal, and characterize its postsynaptic actions. This is important, because different mixtures of cotransmitters, and consequently of postsynaptic action may occur with different presynaptic patterns of activity.

## 5 NEURAL NETWORKS ARE MULTIPLY MODULATED

Neural networks are controlled by many modulatory inputs and substances. Figure 2 illustrates the patterns of modulatory control to the crustacean stomatogastric nervous system, where the motor patterns produced by the only 30 neurons of the stomatogastric ganglion are controlled by about 60 input fibers (Coleman *et al* 1992) that contain at least 15 different substances, including a variety of amines, amino acids, and neuropeptides (Marder and Weimann 1992; Harris-Warrick *et al* 1992). Each of these modulatory substances produces characteristic and different effects on the motor patterns of the STG (Figs. 3,4). This can be understood if one remembers that the intrinsic membrane properties as well as the strengths of the synaptic connections within this group of neurons are all subject to modulation. Because each cell has many conductances, many of which are subject to modulation, and because of the large number of synaptic connections, the modes of circuit operation are theoretically large.

## 6 CIRCUIT RECONFIGURATION BY MODULATORY CONTROL

Figure 3 illustrates that modulatory substances can tune the operation of a single functional circuit. However, neuromodulatory substances can also produce far more extensive changes in the functional organization of neuronal networks. Recent work on the STG demonstrates that sensory and modulatory neurons and substances can cause neurons to switch between different functional circuits, so that the same neuron is part of several different pattern generating circuits at different times (Hooper and Moulins 1989; Dickinson *et al* 1990; Weimann *et al* 1991; Meyrand *et al* 1991; Heinzel et al 1993).

In the example shown in Fig. 4, in control saline the LG neuron is firing in time with the fast pyloric rhythm (the LP neuron is also firing in pyloric time), but there is no ongoing gastric rhythm. When the gastric rhythm was activated by application of the peptide $SDRNFLRF_{NHz}$, the LG neuron fired in time with the gastric rhythm (Weimann *et al* 1993). These and other data lead us to conclude that it is the *modulatory environment that constructs the functional circuit that produces a given behavior* (Meyrand *et al* 1991). Thus, by tuning intrinsic membrane properties and synaptic strengths, neuromodulatory agents can recombine the same neurons into a variety of circuits, capable of generating remarkably distinct outputs.

### Acknowledgements

I thank Dr. Petra Skiebe for Fig 3 art work. Research was supported by NS17813.

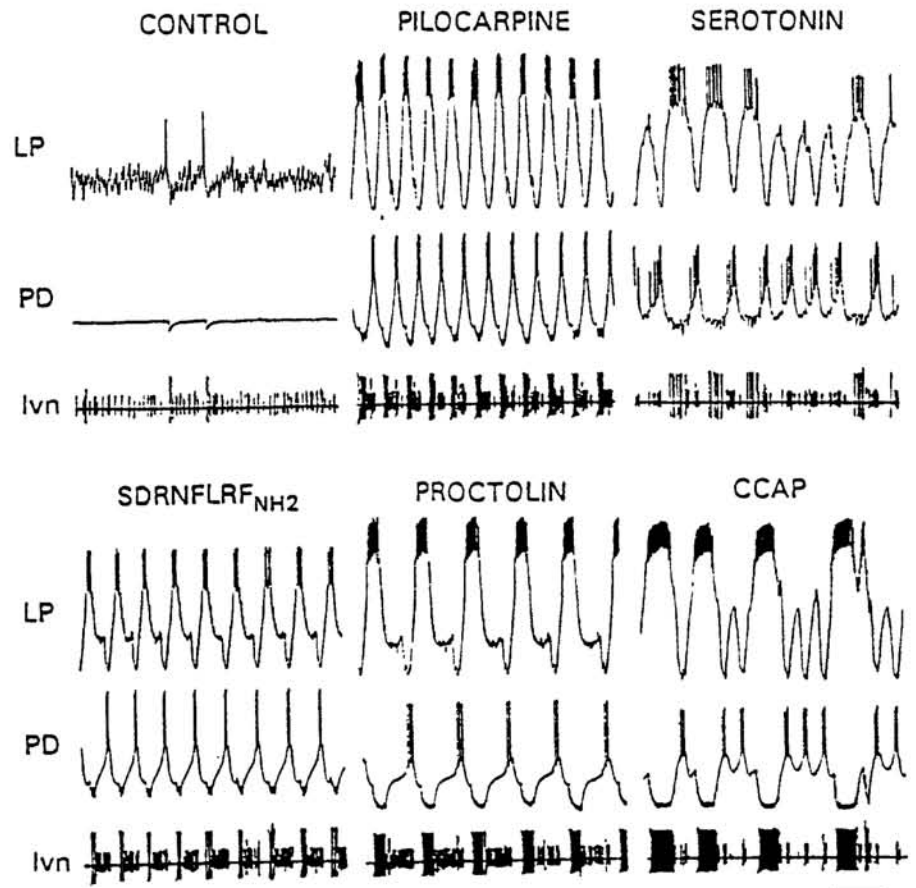

Figure 3: Different forms of the pyloric rhythm different modulators. Each panel, the top two traces: simulataneous intracellular recordings from LP and PD neurons of crab STG; bottom trace: extracellular recording, *lvn* nerve. Control, rhythmic pyloric activity absent. Substances were bath applied, the pyloric patterns produced were different. Modified from Marder and Weimann 1992.

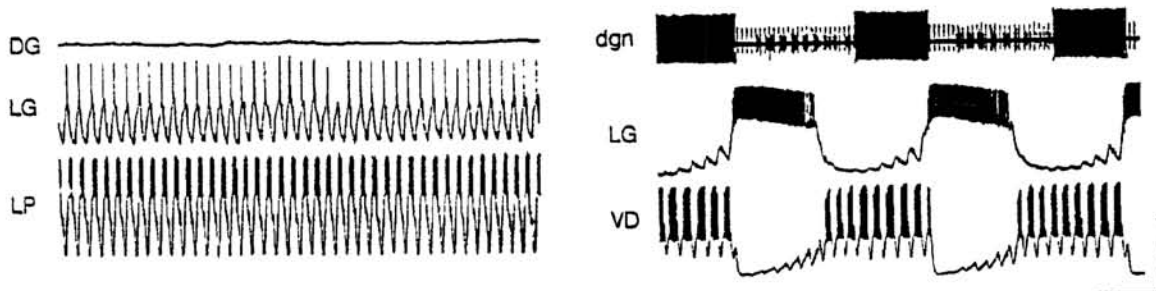

Figure 4: Neurons switch between different pattern-genreating circuits. Left panel, the gastric rhythm not active (monitored by DG neuron), LG neuron in time with the pyloric rhythm (seen as activity in LP neuron). Right panel, gastric rhythm activated by SDRNFLRFamide, monitored by the DG neuron bursts recorded on the *dgn*. LG now fired in alternation with DG neuron. Pyloric time is seen as the interruptions in the activity of the VD neuron. Modified from Marder and Weimann 1992.

## References

Adams WB and Levitan IB 1985 Voltage and ion dependencies of the slow currents which mediate bursting in *Aplysia* neurone $R_{15}$. *J Physiol* **360** 69-93

Clark GA, Kandel ER 1993 Induction of long-term facilitation in *Aplysia* sensory neurons by local application of serotonin to remote synapses. *Proc Natl Acad Sci USA* **90**: 1141-11415

Coleman MJ, Nusbaum MP, Cournil I, Claiborne BJ 1992 Distribution of mopdulatory inputs to the stomatogastric ganglion of the crab, *Cancer borealis*. *J Comp Neur* **325**: 581-594

Dickinson PS, Mecsas C, Marder E 1990 Neuropeptide fusion of two motor-pattern generator circuits. *Nature* **344**: 155-158

Dowling JE 1989 Neuromodulation in the retina: the role of dopamine. *Sem Neur* 1:35-43

Emptage NJ and Carew TJ 1993 Long-term synaptic facilitation in the absence of short-term facilitation in *Aplysia* neurons. *Science* **262** 253-256

Epstein IR, Marder E 1990 Multiple modes of a conditional neural oscillator. *Biol Cybern* **63**: 25-34

Harris-Warrick RM, Flamm RE 1987 Multiple mechanisms of bursting in a conditional bursting neuron. *J Neurosci* **7**: 2113-2128

Harris-Warrick RM, Marder E, Selverston AI, Moulins M eds 1992 **Dynamic Biological Networks: The Stomatogastric Nervous System.** MIT Press Cambridge

Heinzel H-G, Weimann JM, Marder E 1993 The behavioral repertoire of the gastric mill in the crab, *Cancer pagurus*: An *in vivo* endoscopic and electrophysiological examination. *J Neurosci* **13**: 1793-1803

Hooper SL, Moulins M 1989 Switching of a neuron from one network to another by sensory-induced changes in membrane properties. *Science* **244**: 1587-1589

Katz PS, Kirk MD, and Govind CK 1993 Facilitation and depression at different branches of the same motor axon: evidence for presynaptic differences in release. *J Neurosci* **13**: 3075-3089

Kepler TB, Marder E, Abbott LF 1990 The effect of electrical coupling on the frequency of model neuronal oscillators. *Science* **248**: 83-85

Leresche N, Lightowler S, Soltesz I, Jassik-Gerschenfeld D, and Crunelli V 1991 Low-frequency oscillatory activities intrinsic to rat and car thalamocortical cells. *J Physiol* **441** 155-174

Llinás RR 1988 The intrinsic electrophysiological properties of mammalian neurons: insights into central nervous system function. *Science* **242** 1654-1664

McCormick DA 1991 Functional properties of slowly inactivating potassium current in guinea pig dorsal lateral geniculate relay neurons. *J Physiol* **66** 1176-1189

Marom S and Abbott LF 1994 Modeling state-dependent inactivation of membrane currents. *Biophysical J* in press

Marder E, Weimann JM 1992 Modulatory control of multiple task processing in the stomatogastric nervous system. IN: **Neurobiology of Motor Programme Selection: new approaches to mechanisms of behavioral choice,** Kien J,

McCrohan C, Winlow W eds Pergamon Press Oxford

Meyrand P, Simmers J, Moulins, M 1991 Construction of a pattern-generating circuit with neurons of different networks. *Nature* **351**: 60-63

Skinner FK, Turrigiano GG, Marder E 1993 Frequency and burst duration in oscillating neurons and two cell networks. *Biol Cybern* **69**: 375-383

Somers D, Kopell N 1993 Rapid synchronization through fast threshold modulation. *Biol Cybern* **68**: 393-407

Storm JF 1987 Temporal integration by a slowly inactivating K$^+$ current in hippocampal neurons. *Nature* **336**: 379-381

Weimann JM, Meyrand P, Marder E 1991 Neurons that form multiple pattern generators: Identification and multiple activity patterns of gastric/pyloric neurons in the crab stomatogastric system. *J Neurophysiol* **65**: 111-122

Weimann JM, Marder E, Evans B, Calabrese RL 1993 The effects of SDRNFLRF$_{NH2}$ and TNRNFLRF$_{NH2}$ on the motor patterns of the stomatogastric ganglion of the crab, *Cancer borealis*. *J Exp Biol* **181**: 1-26